# NEURAL NETWORK RECOGNIZER FOR HAND-WRITTEN ZIP CODE DIGITS

J. S. Denker, W. R. Gardner, H. P. Graf, D. Henderson, R. E. Howard,
W. Hubbard, L. D. Jackel, H. S. Baird, and I. Guyon
AT&T Bell Laboratories
Holmdel, New Jersey 07733

## ABSTRACT

This paper describes the construction of a system that recognizes hand-printed digits, using a combination of classical techniques and neural-net methods. The system has been trained and tested on real-world data, derived from zipcodes seen on actual U.S. Mail. The system rejects a small percentage of the examples as unclassifiable, and achieves a very low error rate on the remaining examples. The system compares favorably with other state-of-the art recognizers. While some of the methods are specific to this task, it is hoped that many of the techniques will be applicable to a wide range of recognition tasks.

## MOTIVATION

The problem of recognizing hand-written digits is of enormous practical and theoretical interest [Kahan, Pavlidis, and Baird 1987; Watanabe 1985; Pavlidis 1982]. This project has forced us to formulate and deal with a number of questions ranging from the basic psychophysics of human perception to analog integrated circuit design.

This is a topic where "neural net" techniques are expected to be relevant, since the task requires closely mimicking human performance, requires massively parallel processing, involves confident conclusions based on low precision data, and requires learning from examples. It is also a task that can benefit from the high throughput potential of neural network hardware.

Many different techniques were needed. This motivated us to compare various classical techniques as well as modern neural-net techniques. This provided valuable information about the strengths, weaknesses, and range of applicability of the numerous methods.

The overall task is extremely complex, so we have broken it down into a great number of simpler steps. Broadly speaking, the recognizer is divided into the *preprocessor* and the *classifier*. The two main ideas behind the preprocessor are (1) to remove meaningless variations (i.e. noise) and (2) to capture meaningful variations (i.e. salient features).

Most of the results reported in this paper are based on a collection of digits taken from hand-written Zip Codes that appeared on real U.S. Mail passing through the

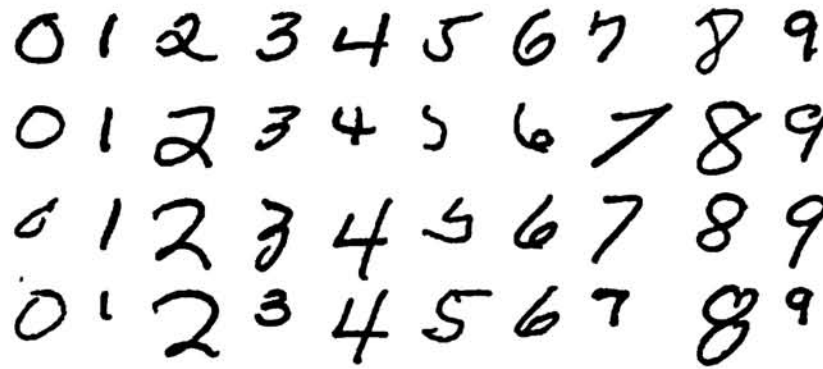

Figure 1: Typical Data

Buffalo, N.Y. post office. Details will be discussed elsewhere [Denker et al., 1989]. Examples of such images are shown in figure 1. The digits were written by many different people, using a great variety of writing styles and instruments, with widely varying levels of care.

Important parts of the task can be handled nicely by our lab's custom analog neural network VLSI chip [Graf et al., 1987; Graf & deVegvar, 1987], allowing us to perform the necessary computations in a reasonable time. Also, since the chip was *not* designed with image processing in mind, this provided a good test of the chips' versatility.

## THE PREPROCESSOR

### Acquisition

The first step is to create a digital version of the image. One must find where on the envelope the zipcode is, which is a hard task in itself [Wang and Srihari 1988]. One must also separate each digit from its neighbors. This would be a relatively simple task if we could assume that a character is contiguous and is disconnected from its neighbors, but neither of these assumptions holds in practice. It is also common to find that there are meaningless stray marks in the image.

Acquisition, binarization, location, and preliminary segmentation were performed by Postal Service contractors. In some images there were extraneous marks, so we developed some simple heuristics to remove them while *preserving*, in most cases, all segments of a split character.

### Scaling and Deskewing

At this point, the size of the image is typically 40 × 60 pixels, although the scaling routine can accept images that are arbitrarily large, or as small as 5 × 13 pixels. A translation and scale factor are then applied to make the image fit in a rectangle

$20 \times 32$ pixels. The character is centered in the rectangle, and just touches either the horizontal or vertical edges, whichever way fits. It is clear that any extraneous marks must be removed before this step, lest the good part of the image be radically compressed in order to make room for some wild mark. The scaling routine changes the horizontal and vertical size of the image by the same factor, so the aspect ratio of the character is preserved.

As shown in figure 1, images can differ greatly in the amount of skew, yet be considered the same digit. This is an extremely significant noise source. To remove this noise, we use the methods of [Casey 1970]; see also [Naylor 1971]. That is, we calculate the $XY$ and $YY$ moments of the image, and apply a linear transformation that drives the $XY$ moment to zero. The transformation is a pure shear, not a rotation, because we find that rotation is much less common than skew.

The operations of scaling and deskewing are performed in a single step. This yields a speed advantage, and, more importantly, eliminates the quantization noise that would be introduced by storing the intermediate images as pixel maps, were the calculation carried out in separate steps.

**Skeletonization**

For the task of digit recognition, the width of the pen used to make the characters is completely meaningless, and is highly variable. It is important to remove this noise source. By deleting pixels at the boundaries of thick strokes. After a few iterations of this process, each stroke will be as thin as possible. The idea is to remove as many pixels as possible without breaking the connectivity. Connectivity is based on the 8 nearest neighbors.

This can be formulated as a pattern matching problem — we search the image looking for situations in which a pixel should be deleted. The decisions can be expressed as a convolution, using a rather small kernel, since the identical decision process is repeated for each location in the image, and the decision depends on the configuration of the pixel's nearest and next-nearest neighbors.

Figure 2 shows an example of a character before ($e$) and after ($f$) skeletonization. It also shows some of the templates we use for skeletonization, together with an indication of where (in the given image) that template was active. To visualize the convolution process, imagine taking a template, laying it over the image in each possible place, and asking if the template is "active" in that place. (The template is the convolution kernel; we use the two terms practically interchangeably.) The portrayal of the template uses the following code: Black indicates that if the corresponding pixel in the image is ON, it will contribute $+1$ to the activity level of this template. Similarly, gray indicates that the corresponding pixel, if ON, will contribute $-5$, reducing the activity of this template. The rest of the pixels don't matter. If the net activity level exceeds a predetermined threshold, the template is considered active at this location. The outputs of all the skeletonizer templates

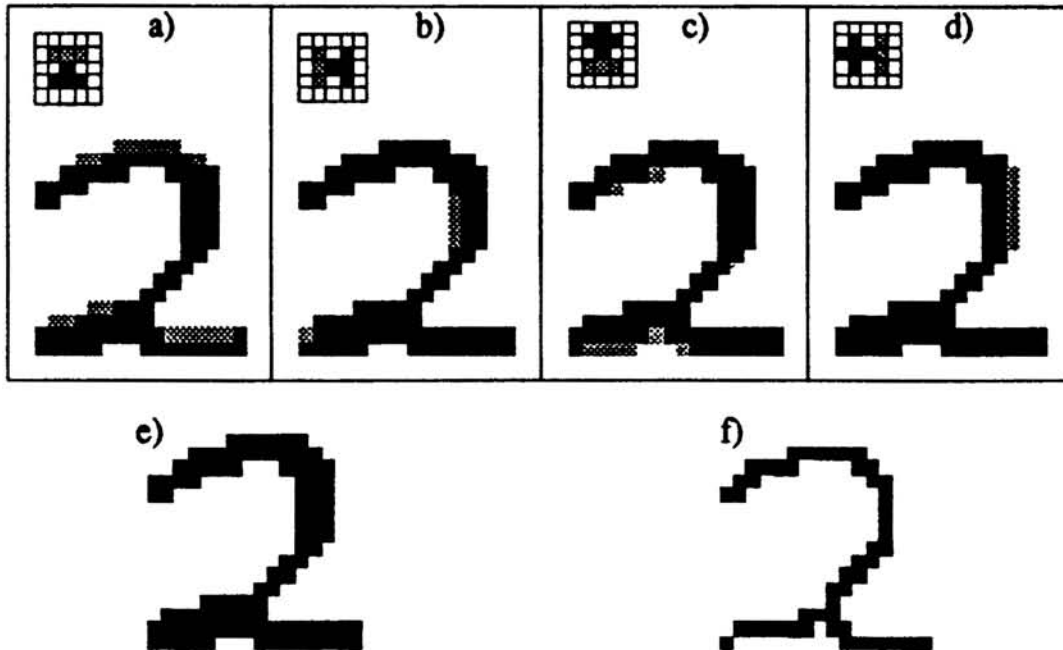

Figure 2: Skeletonization

are combined in a giant logical OR, that is, whenever *any* template is active, we conclude that the pixel presently under the center of the template should be deleted.

The skeletonization computation involves six nested loops:

```
for each iteration I
    for all X in the image (horizontal coordinate)
        for all Y in the image (vertical coordinate)
            for all T in the set of template shapes
                for all P in the template (horizontal)
                    for all Q in the template (vertical)
                        compare image element(X+P, Y+Q)
                        with template(T) element(P, Q)
```

The inner three loops (the loops over $T$, $P$, and $Q$) are performed in parallel, in a single cycle of our special-purpose chip. The outer three loops ($I$, $X$, and $Y$) are performed serially, calling the chip repeatedly. The $X$ and $Y$ loops could be performed in parallel with no change in the algorithms. The additional parallelism would require a proportionate increase in hardware.

The purpose of template *a* is to detect pixels at the top edge of a thick horizontal line. The three "should be OFF" (light grey shade in figure 2) template elements enforce the requirement that this should be a boundary, while the three "should be ON" (solid black shade in figure 2) template elements enforce the requirement that the line be at least two pixels wide.

Template *b* is analogous to template *a*, but rotated 90 degrees. Its purpose is to detect pixels at the left edge of a thick vertical line.

Template *c* is similar to, but not exactly the same as, template *a* rotated 180 degrees. The distinction is necessary because all templates are applied in parallel. A stroke that is only two pixels thick must not be attacked from both sides at once, lest it be removed entirely, changing the connectivity of the image. Previous convolutional line-thinning schemes [Naccache 1984] used templates of size 3 × 3, and therefore had to use several serial sub-stages. For parallel operation *at least* 3 × 4 kernels are needed, and 5 × 5 templates are convenient, powerful, and flexible.

### Feature Maps

Having removed the main sources of meaningless variation, we turn to the task of extracting the meaningful information. It is known from biological studies [Hubel and Wiesel 1962] that the human vision system is sensitive to certain features that occur in images, particularly lines and the ends of lines. We therefore designed detectors for such features. Previous artificial recognizers [Watanabe 1985] have used similar feature extractors.

Once again we use a convolutional method for locating the features of interest — we check each location in the image to see if each particular feature is present there. Figure 3 shows some of the templates we use, and indicates where they become active in an example image. The feature extractor templates are 7 × 7 pixels — slightly larger than the skeletonizer templates.

Feature *b* is designed to detect the right-hand end of (approximately) horizontal strokes. This can be seen as follows: in order for the template to become active at a particular point, the image must be able to touch the "should be ON" pixels at the center of the template without touching the surrounding horseshoe-shaped collection of "'must be OFF" pixels. Essentially the only way this can happen is at the right-hand end of a stroke. (An isolated dot in the image will also activate this template, but the images, at this stage, are not supposed to contain dots). Feature *d* detects (approximately) horizontal strokes.

There are 49 different feature extractor templates. The output of each is stored separately. These outputs are called feature maps, since they show what feature(s) occurred where in the image. It is possible, indeed likely, that several different features will occur in the same place.

Whereas the outputs of all the skeletonizer templates were combined in a very simple way (a giant OR), the outputs of the feature extractor templates are combined in

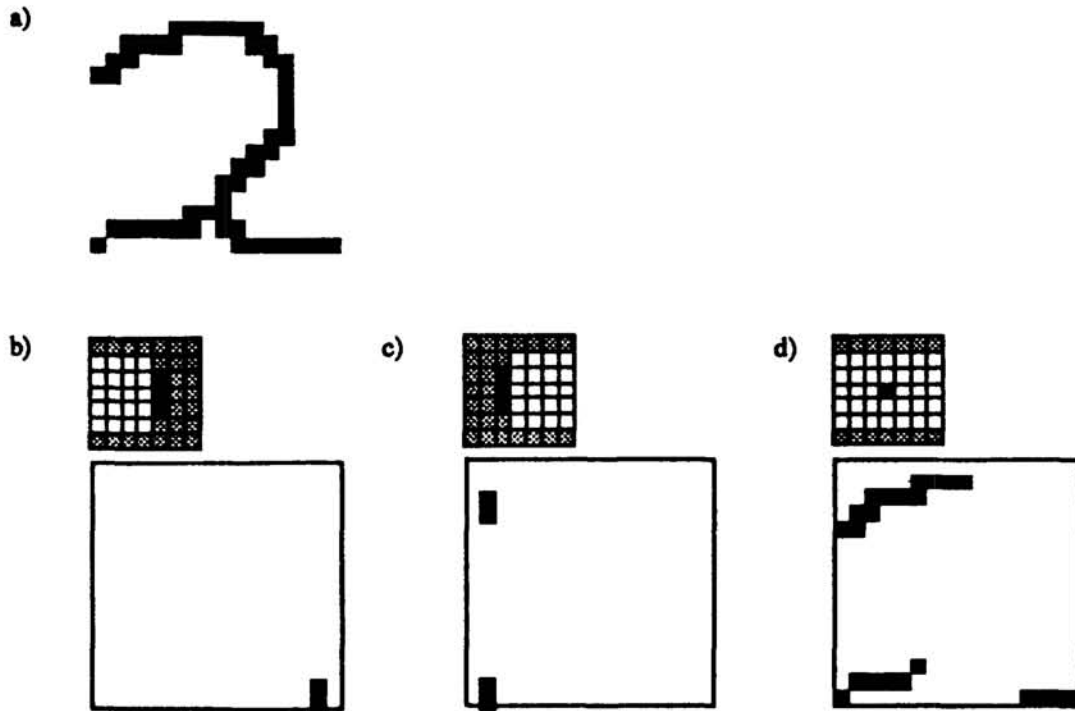

Figure 3: Feature Extraction

various artful ways. For example, feature *b* and a similar one are ORed to form a single combined feature that responds to right-hand ends in general. Certain other features are ANDed to form detectors for arcs (long curved strokes). There are 18 combined features, and these are what is passed to the next stage.

We need to create a compact representation, but starting from the skeletonized image, we have, instead, created 18 feature maps of the same size. Fortunately, we can now return to the theme of removing meaningless variation.

If a certain image contains a particular feature (say a left-hand stroke end) in the upper left corner, it is not really necessary to specify the location of that feature with great precision. To recognize the *shape* of the feature required considerable precision at the input to the convolution, but the *position* of the feature does not require so much precision at the output of the convolution. We call this Coarse Blocking or Coarse Coding of the feature maps. We find that 3 × 5 is sufficent resolution.

## CLASSIFIERS

If the automatic recognizer is unable to classify a particular zip code digit, it may be possible for the Post Office to determine the correct destination by other means. This is costly, but not nearly so costly as a misclassification (substitution error) that causes the envelope to be sent to the wrong destination. Therefore it is critically

important for the system to provide estimates of its confidence, and to reject digits rather than misclassify them.

The objective is not simply to maximize the number of classified digits, nor to minimize the number of errors. The objective is to minimize the cost of the whole operation, and this involves a tradeoff between the rejection rate and the error rate.

## Preliminary Investigations

Several different classifiers were tried, including Parzen Windows, $K$ nearest neighbors, highly customized layered networks, expert systems, matrix associators, feature spins, and adaptive resonance. We performed preliminary studies to identify the most promising methods. We determined that the top three methods in this list were significantly better suited to our task than the others, and we performed systematic comparisons only among those three.

## Classical Clustering Methods

We used two classical clustering techniques, Parzen Windows (PW) and $K$ Nearest Neighbors ($K$NN), which are nicely described in Duda and Hart [1973]. In this application, we found (as expected) that they behaved similarly, although PW consistently outperformed $K$NN by a small margin. These methods have many advantages, not the least of which is that they are well motivated and easily understood in terms of standard Bayesian inference theory. They are well suited to implementation on parallel computers and/or custom hardware. They provide excellent confidence information.

Unlike modern adaptive network methods, PW and $K$NN require no "learning time", Furthermore the performance was reproducible and responded smoothly to improvements in the preprocessor and increases in the size of the training set. This is in contrast to the "noisy" performance of typical layered networks. This is convenient, indeed crucial, during exploratory work.

## Adaptive Network Methods

In the early phases of the project, we found that neural network methods gave rather mediocre results. Later, with a high-performance preprocessor, plus a large training database, we found that a layered network gave the best results, surpassing even Parzen Windows. We used a network with two stages of processing (i.e., two layers of weights), with 40 hidden units and using a one-sided objective function (as opposed to LMS) as described in [Denker and Wittner 1987]. The main theoretical advantage of the layered network over the classical methods is that it can form "higher order" features — conjunctions and disjunctions of the features provided by our feature extractor. Once the network is trained, it has the advantage that the classification of each input is very rapid compared to PW or $K$NN. Furthermore, the weights represent a compact distillation of the training data and thus have a smaller memory requirement. The network provides confidence information that is

just as good as the classical methods. This is obtained by comparing the activation level of the most active output against the runner-up unit(s).

To check on the effectiveness of the preprocessing stages, we applied these three classification schemes (PW, $KNN$, and the two-layer network) on 256-bit vectors consisting of raw bit maps of the images — with no skeletonization and no feature extraction. For each classification scheme, we found the error rate on the raw bit maps was at least a factor of 5 greater than the error rate on the feature vectors, thus clearly demonstrating the utility of feature extraction.

## TESTING

It is impossible to compare the performance of recognition systems except on identical databases. Using highly motivated "friendly" writers, it is possible to get a dataset that is so clean that practically any algorithm would give outstanding results. On the other hand, if the writers are not motivated to write clearly, the result will be not classifiable by machines of any sort (nor by humans for that matter). It would have been much easier to classify digits that were input using a mouse or bitpad, since the lines in the such an image have zero thickness, and stroke-order information is available. It would also have been much easier to recognize digits from a single writer.

The most realistic test data we could obtain was provided by the US Postal Service. It consists of approximately 10,000 digits (1000 in each category) obtained from the zip codes on actual envelopes. The data we received had already been binarized and divided into images of individual digits, rather than multi-digit zip codes, but no further processing had been done.

On this data set, our best performance is as follows: if 14% of the images are rejected as unclassifiable, only 1% of the remainder are misclassified. If *no* images are rejected, approximately 6% are misclassified. Other groups are working with the same dataset, but their results have not yet been published. Informal communications indicate that our results are among the best.

## CONCLUSIONS

We have obtained very good results on this very difficult task. Our methods include low-precision and analog processing, massively parallel computation, extraction of biologically-motivated features, and learning from examples. We feel that this is, therefore, a fine example of a Neural Information Processing System. We emphasize that old-fashioned engineering, classical pattern recognition, and the latest learning-from-examples methods were all absolutely necessary. Without the careful engineering, a direct adaptive network attack would not succeed, but by the same token, without learning from a very large database, it would have been excruciating to engineer a sufficiently accurate representation of the probability space.

## Acknowledgements
It is a pleasure to acknowledge useful discussions with Patrick Gallinari and technical assistance from Roger Epworth. We thank Tim Barnum of the U.S. Postal Service for making the Zip Code data available to us.

## References

1. R. G. Casey, "Moment Normalization of Handprinted Characters", IBM J. Res. Develop., 548 (1970)

2. J. S. Denker et al., "Details of the Hand-Written Character Recognizer", to be published (1989)

3. R. O. Duda and P. E. Hart, **Pattern Classification and Scene Analysis**, John Wiley and Sons (1973)

4. E. Gullichsen and E. Chang, "Pattern Classification by Neural Network: An Experimental System for Icon Recognition", Proc. IEEE First Int. Conf. on Neural Networks, San Diego, **IV**, 725 (1987)

5. H. P. Graf, W. Hubbard, L. D. Jackel, P.G.N. deVegvar, "A CMOS Associative Memory Chip", Proc. IEEE First Int. Conf. on Neural Networks, San Diego, **III**,461 (1987)

6. H.P Graf and P. deVegvar, "A CMOS Implementation of a Neural Network Model", Proc. 1987 Stanford Conf. Advanced Res. VLSI, P. Losleben (ed.) MIT Press, 351 (1987)

7. D. H. Hubel and T. N. Wiesel, "Receptive fields, binocular interaction and functional architecture in the cat's visual cortex", J. Physiology **160**, 106 (1962)

8. S. Kahan, T. Pavlidis, and H. S. Baird, "On the Recognition of Printed Characters of Any Font and Size", IEEE Transactions on Pattern Analysis and Machine Intelligence, **PAMI-9**, 274 (1987)

9. N. J. Naccache and R. Shinghal, "SPTA: A Proposed Algorithm for Thinning Binary Patterns", IEEE Trans. Systems, Man, and Cybernetics, **SMC-14**, 409 (1984)

10. W. C. Naylor, "Some Studies in the Interactive Design of Character Recognition Systems", IEEE Transactions on Computers, 1075 (1971)

11. T. Pavlidis, **Algorithms for Graphics and Image Processing**, Computer Science Press (1982)

12. C. Y. Suen, M. Berthod, and S. Mori, "Automatic Recognition of Handprinted Characters — The State of the Art", Proceedings of the IEEE 68 4, 469 (1980).

13. C-H. Wang and S. N. Srihari, "A Framework for Object Recognition in a Visually Complex Environment and its Application to Locating Address Blocks on Mail Pieces", Intl. J. Computer Vision **2**, 125 (1988)

14. S. Watanabe, **Pattern Recognition**, John Wiley and Sons, New York (1985)